# Conditional Models on the Ranking Poset

**Guy Lebanon**
School of Computer Science
Carnegie Mellon University
Pittsburgh, PA 15213
lebanon@cs.cmu.edu

**John Lafferty**
School of Computer Science
Carnegie Mellon University
Pittsburgh, PA 15213
lafferty@cs.cmu.edu

## Abstract

A distance-based conditional model on the ranking poset is presented
for use in classification and ranking. The model is an extension of the
Mallows $\phi$ model, and generalizes the classifier combination methods
used by several ensemble learning algorithms, including error correcting
output codes, discrete AdaBoost, logistic regression and cranking. The
algebraic structure of the ranking poset leads to a simple Bayesian inter-
pretation of the conditional model and its special cases. In addition to a
unifying view, the framework suggests a probabilistic interpretation for
error correcting output codes and an extension beyond the binary coding
scheme.

## 1 Introduction

Classification is the task of associating a single label $y \in \mathcal{Y}$ with a covariate $x$. A gener-
alization of this problem is *conditional ranking*, the task of assigning to $x$ a full or partial
ranking of the items in $\mathcal{Y}$. This paper studies the algebraic structure of this problem, and
proposes a combinatorial structure called the ranking poset for building probability models
for conditional ranking.

In ensemble approaches to classification and ranking, several base models are combined to
produce a single ranker or classifier. An important distinction between different ensemble
methods is whether they use discrete inputs, ranked inputs, or confidence-rated predic-
tions. In the case of discrete inputs, the base models provide a single item in $\mathcal{Y}$, and no
preference for a second or third choice is given. In the case of ranked input, the base clas-
sifiers output a full or partial ranking over $\mathcal{Y}$. Of course, discrete input is a special case
of ranked input, where the partial ranking consists of the single topmost item. In the case
of confidence-rated predictions, the base models again output full or partial rankings, but
in addition provide a confidence score, indicating how much one class should be preferred
to another. While confidence-rated predictions are sometimes preferable as input to an en-
semble method, such confidence scores are often not available (as is typically the case in
metasearch), and even when they are available, the scores may not be well calibrated.

This paper investigates a unifying algebraic framework for ensemble methods for clas-
sification and conditional ranking, focusing on the cases of discrete and ranked inputs.
Our approach is based on the ranking poset on $n$ items, denoted $\mathfrak{W}_n$, which consists of
the collection of all full and partial rankings equipped with the partial order given by re-

finement of rankings. The structure of the poset of partial ranking over $\mathcal{Y}$ gives rise to natural invariant distance functions that generalize Kendall's Tau and the Hamming distance. Using these distance functions we define a conditional model $p_\theta(\pi \mid \sigma_1, \ldots, \sigma_k)$ where $\pi, \sigma_1, \ldots, \sigma_k \in \mathfrak{W}_n$. This conditional model generalizes several existing models for classification and ranking, and includes as a special case the Mallows $\phi$ model [11]. In addition, the model represents algebraically the way in which input classifiers are combined in certain ensemble methods, including error correcting output codes [4], several versions of AdaBoost [7, 1], and cranking [10].

In Section 2 we review some basic algebraic concepts and in Section 3 we define the ranking poset. The new model and its Bayesian interpretation are described in Section 4. A derivation of some special cases is given in Section 5, and we conclude with a summary in Section 6.

## 2  Permutations and Cosets

We begin by reviewing some basic concepts from algebra, with some of the notation and definitions borrowed from Critchlow [2].

Identifying the items to be ranked $y_1, \ldots, y_n$ with the numbers $1, \ldots, n$, if $\pi$ denotes a permutation of $\{1, \ldots, n\}$, then $\pi(i)$ denotes the rank given to item $i$ and $\pi^{-1}(i)$ denotes the item assigned to rank $i$. The collection of all permutations of $n$-items forms the non-abelian *symmetric group of order $n$*, denoted $\mathfrak{S}_n$. The multiplicative notation $\pi \cdot \sigma = \pi\sigma$ is used to denote function composition.

The subgroup of $\mathfrak{S}_n$ consisting of all permutations that fix the top $k$ positions is denoted $\mathfrak{S}_{n-k}$; thus,

$$\mathfrak{S}_{n-k} = \{\pi \in \mathfrak{S}_n \mid \pi(i) = i, \ i = 1, \ldots, k\}. \tag{1}$$

The right coset

$$\mathfrak{S}_{n-k}\pi = \{\sigma\pi \mid \sigma \in \mathfrak{S}_{n-k}\} \tag{2}$$

is equivalent to a partial ranking, where there is a full ordering of the $k$ top-ranked items. The set of all such partial rankings forms the quotient space $\mathfrak{S}_n / \mathfrak{S}_{n-k}$.

An *ordered partition of $n$* is a sequence $\gamma = n_1, \ldots, n_r$ of positive integers that sum to $n$. Such an ordered partition corresponds to a partial ranking of type $\gamma$ with $n_1$ items in the first position, $n_2$ items in the second position and so on. No further information is conveyed about orderings within each position. A partial ranking of the top $k$ items is a special case with $r = k+1, n_1 = \ldots = n_k = 1, n_{k+1} = n - k$. More formally, let $N_1 = \{1, \ldots, n_1\}, N_2 = \{n_1+1, \ldots, n_1+n_2\}, \cdots, N_r = \{n_1+\cdots+n_{r-1}+1, \ldots, n\}$. Then the subgroup $\mathfrak{S}_\gamma \cong \mathfrak{S}_{n_1} \times \cdots \times \mathfrak{S}_{n_r}$ contains all permutations $\pi \in \mathfrak{S}_n$ for which the set equality $\pi(N_i) = N_i$ holds for each $i$; that is, all permutations that only permute within each $N_i$. A partial ranking of type $\gamma$ is equivalent to a coset $\mathfrak{S}_\gamma\pi$ and the set of such partial rankings forms the quotient space $\mathfrak{S}_n / \mathfrak{S}_\gamma$.

We now describe a convenient notation for permutations and cosets. In the following, we list items separated by vertical lines, indicating that the items on the left side of the line are preferred to (ranked higher than) the items on the right side of the line. For example, the permutation $\pi(1) = 2, \pi(2) = 1, \pi(3) = 3$ is denoted by $2|1|3$. A partial ranking $\mathfrak{S}_{5-3}\pi$ where the top 3 items are $3, 2, 1$ is denoted by $3|2|1|4, 5$. A classification $y_3$ may thus be denoted by $3|1, 2, 4, 5$. A partial ranking $\mathfrak{S}_\gamma$ where $\gamma = 3, 2$ with items $1, 3, 5$ ranked in the first position is denoted by $1, 3, 5|2, 4$.

A distance function $d$ on $\mathfrak{S}_n$ is a function $d : \mathfrak{S}_n \times \mathfrak{S}_n \to \mathbb{R}$ that satisfies the usual properties: $d(\pi, \pi) = 0$, $d(\pi, \sigma) > 0$ when $\pi \neq \sigma$, $d(\pi, \sigma) = d(\sigma, \pi)$, and the triangle

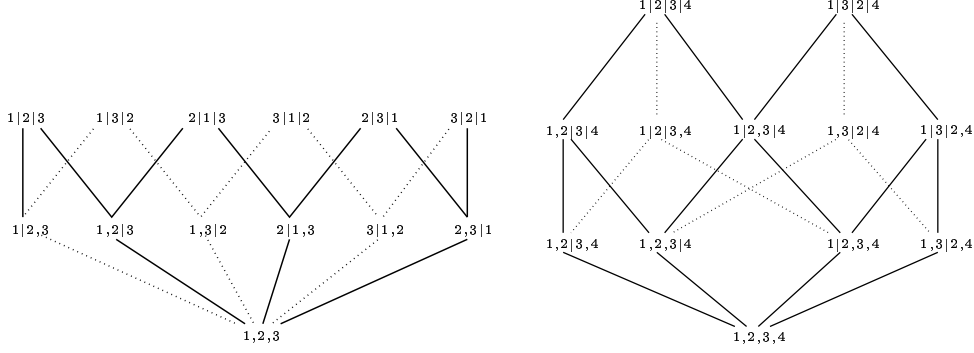

Figure 1: The Hasse diagram of $\mathfrak{W}_3$ (left) and a partial Hasse diagram of $\mathfrak{W}_4$ (right). Some of the lines are dotted for easier visualization.

inequality $d(\pi, \sigma) \le d(\pi, \tau) + d(\tau, \sigma)$ for all $\pi, \sigma, \tau \in \mathfrak{S}_n$. In addition, since the indexing of the items $y_1, \ldots, y_n$ is arbitrary, it is appropriate to require invariance to relabeling of $\mathcal{Y}$. Formally, this amounts to right invariance $d(\pi, \sigma) = d(\pi\tau, \sigma\tau)$, for all $\pi, \sigma, \tau \in \mathfrak{S}_n$.

A popular right invariant distance on $\mathfrak{S}_n$ is Kendall's Tau $T(\pi, \sigma)$, given by

$$T(\pi, \sigma) \quad = \quad \sum_{i=1}^{n-1} \sum_{l > i} I(\pi\sigma^{-1}(i) - \pi\sigma^{-1}(l)) \tag{3}$$

where $I(x) = 1$ for $x > 0$ and $I(x) = 0$ otherwise [8]. Kendall's Tau $T(\pi, \sigma)$ can be interpreted as the number of discordant pairs of items between $\pi$ and $\sigma$, or the minimum number of adjacent transpositions needed to bring $\pi^{-1}$ to $\sigma^{-1}$. An adjacent transposition flips a pair of items that have adjacent ranks. Critchlow [2] derives extensions of Kendall's Tau and other distances on $\mathfrak{S}_n$ to distances on partial rankings.

## 3  The Ranking Poset

We first define partially ordered sets and then proceed to define the ranking poset. Some of the definitions below are taken from [12], where a thorough introduction to posets can be found.

A partially ordered set or *poset* is a pair $(P, \preceq)$, where $P$ is a set and $\preceq$ is a binary relation that satisfies (1) $x \preceq x$, (2) if $x \preceq y$ and $y \preceq x$ then $x = y$, and (3) if $x \preceq y$ and $y \preceq z$ then $x \preceq z$ for all $x, y, z \in P$. We write $x \prec y$ when $x \preceq y$ and $x \ne y$. We say that $y$ *covers* $x$ and write $x \dashv y$ when $x \prec y$ and there is no $z \in P$ such that $x \prec z$ and $z \prec y$. A finite poset is completely described by the covering relation. The planar Hasse diagram of $(P, \preceq)$ is the graph for which the elements of $P$ are the nodes and the edges are given by the covering relation. In addition, we require that if $x \dashv y$ then $y$ is drawn higher than $x$.

The *ranking poset* $\mathfrak{W}_n$ is the poset in which the elements are all possible cosets $\mathfrak{S}_\lambda \pi$, where $\lambda$ is an ordered partition of $n$ and $\pi \in \mathfrak{S}_n$. The partial order of $\mathfrak{W}_n$ is defined by refinement; that is, $\pi \prec \sigma$ if we can get from $\pi$ to $\sigma$ by adding vertical lines. Note that $\mathfrak{W}_n$ is different from the poset of all set partitions of $\{1, \ldots, n\}$ ordered by partition refinement since in $\mathfrak{W}_n$ the order of the partition elements matters. Figure 1 shows the Hasse diagram of $\mathfrak{W}_3$ and a portion of the Hasse diagram of $\mathfrak{W}_4$.

A *subposet* $(Q, \preceq_Q)$ of $(P, \preceq_P)$ is defined by $Q \subset P$ and $x \preceq_Q y$ if and only if $x \preceq_P y$. A *chain* is a poset in which every two elements are comparable. A *saturated chain* $C$ of

length $k$ is a sequence of elements $x_0, \dots, x_k \in P$ that satisfy $x_0 \dashv x_1 \dashv \cdots \dashv x_k$. A chain of $P$ is a *maximal chain* if there is no other saturated chain of $P$ that contains it. A *graded poset of rank $n$* is a poset in which every maximal chain has length $n$. In a graded poset, there is a *rank* or *grade function* $\rho : P \to \{0, \dots, q\}$ such that $\rho(x) = 0$ if $x$ is a minimal element and $\rho(y) = \rho(x) + 1$ if $x \dashv y$.

It is easy to see that $\mathfrak{W}_n$ is a graded poset of rank $n - 1$ and the rank of every element is the number of vertical lines in its denotation. We use $\mathfrak{W}_{n|A}$ to denote the subposet of $\mathfrak{W}_n$ consisting of $\{x \in \mathfrak{W}_n \mid \rho(x) \in A\}$. In particular, the elements in the $k$th grade, all of which are incomparable, are denoted by $\mathfrak{W}_{n|k}$. Full orderings occupy the topmost grade $\mathfrak{W}_{n|n-1}$. Classifications $i | \{1, \dots, n\} \backslash i$ reside in $\mathfrak{W}_{n|1}$. Other elements of $\mathfrak{W}_{n|1}$ are multilabel classifications $X | \{1, \dots, n\} \backslash X$ where $X \subset \{1, \dots, n\}$.

## 4 Conditional Models on the Ranking Poset

We now present a family of conditional models defined in terms of the ranking poset. To begin, suppose that $d$ is a right invariant function on $\mathfrak{W}_n$. That is, $d(\pi, \sigma) = d(\pi\tau, \sigma\tau)$ for all $\pi, \sigma \in \mathfrak{W}_n$ and $\tau \in \mathfrak{S}_n$. Here right invariance is defined with respect to the natural action of $\mathfrak{S}_n$ on $\mathfrak{W}_n$, given by

$$\{y_{i_1}\}_{I_1} | \{y_{i_2}\}_{I_2} | \cdots | \{y_{i_k}\}_{I_k} \cdot \tau \;=\; \{\tau(y_{i_1})\}_{I_1} | \{\tau(y_{i_2})\}_{I_2} | \cdots | \{\tau(y_{i_k})\}_{I_k}. \qquad (4)$$

The function $d$ may or may not be a metric; its interpretation as a measure of dissimilarity, however, remains.

We will examine several distances that are based on the covering relation $\dashv$ of $\mathfrak{W}_n$. Down and up moves on the Hasse diagram will be denoted by $\searrow$ and $\nearrow$ respectively. A distance $d$ defined in terms of $\searrow$ and $\nearrow$ moves is easily shown to be right invariant because the group action of $\mathfrak{S}_n$ does not change the covering relation between any two elements; that is, the group action of $\mathfrak{S}_n$ on $\mathfrak{W}_n$ commutes with $\searrow$ and $\nearrow$ moves:

$$
\begin{array}{ccc}
\mathfrak{W}_n \xrightarrow{\;\mathfrak{S}_n\;} \mathfrak{W}_n & \qquad & \mathfrak{W}_n \xrightarrow{\;\mathfrak{S}_n\;} \mathfrak{W}_n \\
\searrow\big\downarrow \qquad \big\downarrow\searrow & & \nearrow\big\downarrow \qquad \big\downarrow\nearrow \\
\mathfrak{W}_n \xrightarrow{\;\mathfrak{S}_n\;} \mathfrak{W}_n & & \mathfrak{W}_n \xrightarrow{\;\mathfrak{S}_n\;} \mathfrak{W}_n
\end{array} \qquad (5)
$$

While the metric properties of $d$ are not required in our model, the right invariance property is essential since we want to treat all $y_i$ in the same manner.

We are now ready to give the general form of a conditional model on $\mathfrak{W}_n$. Let $d$ be an invariant function, as above. The model takes as input $k$ rankings $\sigma_1, \sigma_2, \dots, \sigma_k$ contained in some subset $\Sigma \subset \mathfrak{W}_n$ of the ranking poset. For example, each $\sigma_j$ could be an element of $\mathfrak{W}_{n|n-1}$. Let $q_0$ be a probability mass function on $\mathfrak{W}_n$, which will be the "carrier density" or default model. Then $d$ and $q_0$ specify an exponential model $p_\theta(\pi \mid \boldsymbol{\sigma})$ given by

$$p_\theta(\pi \mid \boldsymbol{\sigma}) \;=\; \frac{1}{Z(\boldsymbol{\theta}, \boldsymbol{\sigma})} \, q_0(\pi) \exp\left( \sum_{j=1}^{k} \theta_j \, d(\pi, \sigma_j) \right) \qquad (6)$$

where $\boldsymbol{\theta} \in \Theta \subseteq \mathbb{R}^k$, $\pi \in \Pi \subset \mathfrak{W}_n$, and $\sigma_j \in \Sigma \subset \mathfrak{W}_n$. The term $Z(\boldsymbol{\theta}, \boldsymbol{\sigma})$ is the normalizing constant

$$Z(\boldsymbol{\theta}, \boldsymbol{\sigma}) \;=\; \sum_{\pi \in \Pi} q_0(\pi) \exp\left( \sum_{j=1}^{k} \theta_j \, d(\pi, \sigma_j) \right) . \qquad (7)$$

Thus, conditional on $\boldsymbol{\sigma} \in \Sigma^k$, $p_\theta(\cdot \mid \boldsymbol{\sigma})$ forms a probability distribution over $\pi \in \Pi \subset \mathfrak{W}_n$.

Given a data set $\{(\pi^{(i)}, \boldsymbol{\sigma}^{(i)})\}$, the parameters $\theta \in \Theta$ will typically be selected by maximizing the conditional loglikelihood $\ell(\boldsymbol{\theta}) = \sum_i \log p_\theta(\pi^{(i)} \mid \boldsymbol{\sigma}^{(i)})$, a marginal likelihood or posterior. Under mild regularity conditions, $\ell(\theta)$ will be convex and have a unique global maximum.

## 4.1 A Bayesian interpretation

We now derive a Bayesian interpretation for the model given by (6). Our result parallels the interpretation of multistage ranking models given by Fligner and Verducci [6]. The key fact is that, under appropriate assumptions, the normalizing term does not depend on the partial ordering in the one-dimensional case.

**Proposition 4.1.** *Suppose that $d$ is right invariant and that $\Sigma$ is invariant under the action of $\mathfrak{S}_n$. If $\mathfrak{S}_n$ acts transitively on $\Pi$ then*

$$\sum_{\sigma \in \Sigma} e^{\theta d(\pi, \sigma)} = \sum_{\sigma \in \Sigma} e^{\theta d(\pi', \sigma)} \tag{8}$$

*for all $\pi, \pi' \in \Pi$ and $\theta \in \mathbb{R}$.*

*Proof.* First, note that since $\Sigma \subset \mathfrak{W}_n$ is invariant under the action of $\mathfrak{S}_n$, it follows that $\Sigma\tau = \Sigma$ for each $\tau \in \mathfrak{S}_n$. Indeed, $\Sigma\tau \subseteq \Sigma$ by the invariance assumption, and $\Sigma \subseteq \Sigma\tau$ since for $\sigma = \{y_{i_1}\} \mid \cdots \mid \{y_{i_l}\} \in \Sigma$ we have $\sigma' = \{\tau^{-1}(y_{i_1})\} \mid \cdots \mid \{\tau^{-1}(y_{i_l})\} \in \Sigma$ such that $\sigma'\tau = \sigma$.

Now, since $\mathfrak{S}_n$ acts transitively on $\Pi$, for all $\pi, \pi' \in \Pi$ there is $\eta \in \mathfrak{S}_n$ such that $\pi\eta = \pi'$. We thus have that

$$Z(\theta, \pi) = \sum_{\sigma \in \Sigma} e^{\theta d(\pi, \sigma)} \tag{9}$$

$$= \sum_{\sigma \in \Sigma} e^{\theta d(\pi\eta, \sigma\eta)} \quad \text{(by right invariance of } d\text{)} \tag{10}$$

$$= \sum_{\sigma \in \Sigma} e^{\theta d(\pi', \sigma\eta)} \tag{11}$$

$$= \sum_{\sigma \in \Sigma} e^{\theta d(\pi', \sigma)} \quad \text{(by invariance of } \Sigma\text{)} \tag{12}$$

Thus, we can write $Z(\theta, \pi) = Z(\theta)$ since the normalizing constant for $\pi \in \Pi$ does not in fact depend on $\pi$. $\qquad \square$

The underlying generative model is given as follows. Assume that $\pi \in \Pi$ is drawn from the prior $q_0(\pi)$ and that $\sigma_1, \ldots, \sigma_k$ are independently drawn from generalized Mallows models

$$p_{\theta_j}(\sigma_j \mid \pi) = \frac{1}{Z(\theta_j)} e^{\theta_j d(\pi, \sigma_j)} \tag{13}$$

where $\sigma_j \in \Sigma$. Then under the conditions of Proposition 4.1, we have from Bayes' rule that the posterior distribution over $\pi$ is given by

$$\frac{q_0(\pi) \prod_j p_{\theta_j}(\sigma_j \mid \pi)}{\sum_{\pi \in \Pi} q_0(\pi) \prod_j p_{\theta_j}(\sigma_j \mid \pi)} = \frac{q_0(\pi) e^{\sum_j \theta_j d(\pi, \theta_j)} \prod_j Z(\theta_j)^{-1}}{\prod_j Z(\theta_j)^{-1} \sum_{\pi \in \Pi} q_0(\pi) e^{\sum_j \theta_j d(\pi, \theta_j)}} \tag{14}$$

$$= p_\theta(\pi \mid \boldsymbol{\sigma}) \tag{15}$$

We thus have the following characterization of $p_\theta(\cdot \,|\, \sigma)$.

**Proposition 4.2.** *If $d$ is right invariant, $\Sigma$ is invariant under the action of $\mathfrak{S}_n$, and $\mathfrak{S}_n$ acts transitively on $\Pi$, then the model $p_\theta(\cdot \,|\, \sigma)$ defined in equation* (6) *is the posterior under independent sampling of generalized Mallows models, $\sigma_j \sim p_{\theta_j}(\cdot \,|\, \pi)$, with prior $\pi \sim q_0$.*

The conditions of this proposition are satisfied, for example, when $\Pi = \mathfrak{S}_n/\mathfrak{S}_\lambda$ and $\Sigma = \cup_i \mathfrak{S}_n/\mathfrak{S}_{\lambda_i}$ as is assumed in the special cases of the next section.

## 5 Special Cases

This section derives several special cases of model (6), corresponding to existing ensemble methods. The special cases correspond to different choices of $\Pi, \Sigma, \Theta$ and $d$ in the definition of the model. In each case $q_0(\pi)$ is taken to be uniform, though the extension to non-uniform $q_0(\pi)$ is immediate. Following [9], the unnormalized versions of all the models may be easily derived, corresponding to the exponential loss used in boosting.

### 5.1 Cranking and Mallows $\phi$ model

Let $\Theta = \mathbb{R}^k, \Pi = \Sigma = \mathfrak{W}_{n|n-1} = \mathfrak{S}_n$, and let $d(\pi, \sigma)$ be the minimum number of down-up $(\searrow \nearrow)$ moves on the Hasse diagram of $\mathfrak{W}_n$ needed to bring $\pi$ to $\sigma$. Since adjacent transpositions of permutations may be identified with a down move followed by an up move over the Hasse diagram, $d(\pi, \sigma)$ is equal to Kendall's Tau $T(\pi, \sigma)$. For example, $T(1|2|3, 3|2|1) = 3$ and the corresponding path in Figure 1 is

$$1|2|3 \searrow 1,2|3 \nearrow 2|1|3 \searrow 2|1,3 \nearrow 2|3|1 \searrow 2,3|1 \nearrow 3|2|1.$$

In this case model (6) becomes the cranking model [10]

$$p_\theta(\pi \,|\, \boldsymbol{\theta}) \;=\; \frac{1}{Z(\boldsymbol{\sigma}, \boldsymbol{\theta})} \, e^{\sum_{j=1}^k \theta_j T(\pi, \sigma_j)}, \quad \boldsymbol{\theta} \in \mathbb{R}^k \quad \pi, \sigma_j \in \mathfrak{S}_n. \tag{16}$$

The Bayesian interpretation in this case is well known, and is derived in [6]. The generative model is independent sampling of $\sigma_j$ from a Mallows $\phi$ model whose location parameter is $\pi$ and whose scale parameter is $\theta_j$. Other special cases that fall into this category are the models of Feigin [5] and Critchlow and Verducci [3].

### 5.2 Logistic models

Let $\Theta = \mathbb{R}^k, \Pi = \Sigma = \mathfrak{S}_n/\mathfrak{S}_{n-1}$, and let $d(\pi, \sigma)$ be the minimum number of up-down $(\nearrow \searrow)$ moves in the Hasse diagram. Since $\Pi = \Sigma = \mathfrak{S}_n/\mathfrak{S}_{n-1}$

$$d(\pi, \sigma) = d(\mathfrak{S}_{n-1}\tau, \mathfrak{S}_{n-1}\eta) = \begin{cases} 0 & \text{if } \tau^{-1}(1) = \eta^{-1}(1) \\ 2 & \text{otherwise.} \end{cases} \tag{17}$$

In this case model (6) becomes equivalent to the multiclass generalization of logistic regression. If the normalization constraints in the corresponding convex primal problem are removed, the model becomes discrete AdaBoost.M2; that is, $d(\pi, \sigma_j(x))/2$ becomes the (discrete) multiclass weak learner $f_i(x, y) \in \{0, 1\}$ in the usual boosting notation. See [9] for details on the correspondence between exponential models and the unnormalized models that correspond to AdaBoost.

### 5.3 Error correcting output codes

A more interesting special case of the algebraic structure described in Sections 3 and 4 is where the ensemble method is error correcting output coding (ECOC) [4]. Here we set

$\Pi = \mathfrak{S}_n/\mathfrak{S}_{n-1}$, $\Sigma = \mathfrak{W}_{n|1}\backslash\Pi$, and take the parameter space to be

$$\Theta \;=\; \{\boldsymbol{\theta} \in \mathbb{R}^k \mid \theta_1 = \theta_2 = \cdots = \theta_k, \text{ and } \theta_i < 0\}. \tag{18}$$

As before, $d(\pi, \sigma)$ is the minimal number of up-down ($\nearrow\searrow$) moves in the Hasse diagram needed to bring $\pi$ to $\sigma$.

Since $\pi = \mathfrak{S}_{n-1}\tau$, the model computes probabilities of classifications $y_{\tau^{-1}(1)}$. On input $x$, the base rankers output $\sigma_j(x) \in \mathfrak{W}_{n|1}\backslash\Pi$, which corresponds to one of the binary classifiers in ECOC for the appropriate column of the binary coding matrix. For example, consider a binary classifier trained on the coding column $(1, 0, 1, 0, 0, 0)^\top$. On an input $x$, the classifier outputs 0 or 1, corresponding to the partial rankings $\sigma = 2, 4, 5, 6|1, 3$ and $\sigma = 1, 3|2, 4, 5, 6$, respectively.

Since $\pi \in \mathfrak{S}_n/\mathfrak{S}_{n-1}$ and $\sigma \in \mathfrak{W}_{n|1}\backslash\Pi$

$$\begin{aligned}
d(\pi, \sigma) &= d(\mathfrak{S}_{n-1}\tau, \{y_i\}_{i \in I}|\{y_i\}_{i \in \bar{I}}) \tag{19} \\
&= \begin{cases} 1 & \text{if } \tau^{-1}(1) \in I \\ 2 & \text{otherwise.} \end{cases} \tag{20}
\end{aligned}$$

For example, if $\pi = 2|1, 3, 4, 5, 6$ and $\sigma = 2, 4, 5, 6|1, 3$, then $d(\pi, \sigma) = 1$, as can be seen from the sequence of moves

$$2|1, 3, 4, 5, 6 \;\nearrow\; 2|4, 5, 6|1, 3 \;\searrow\; 2, 4, 5, 6|1, 3\,. \tag{21}$$

If $\pi = 1|2, 3, 4, 5, 6$ and $\sigma = 2, 4, 5, 6|1, 3$, then $d(\pi, \sigma) = 2$, with the sequence of moves

$$1|2, 3, 4, 5, 6 \;\nearrow\; 1|2, 4, 5, 6|3 \;\searrow\; 1, 2, 4, 5, 6|3 \;\nearrow\; 2, 4, 5, 6|1|3 \;\searrow\; 2, 4, 5, 6|1, 3\,. \tag{22}$$

Since $\theta_i = \theta_j$, the exponent of the model becomes $\theta \sum_j d(\pi, \sigma_j)$. At test time, the model thus selects the label corresponding to the partial ranking $\pi^* = \arg\max_\pi p_\theta(\pi \mid \boldsymbol{\sigma})$. Now, since $\theta$ is strictly negative, $p_\theta(\pi \mid \boldsymbol{\sigma})$ is a monotonically decreasing function in $\sum_j d(\pi, \sigma_j)$. Equivalence with the ECOC decision rule thus follows from the fact that $\sum_{j=1}^k d(\pi, \sigma_j) - k$ is the Hamming distance between the appropriate row of the coding matrix and the concatenation of the bits returned from the binary classifiers.

Thus, with the appropriate definitions of $\Pi$, $\Sigma$ and $d$, the conditional model on the ranking poset is a probabilistic formulation of ECOC that yields the same classification decisions. This suggests ways in which ECOC might be naturally extended. First, relaxing the constraint $\theta_1 = \theta_2 = \cdots = \theta_k$ results in a more general model that corresponds to ECOC with a weighted Hamming distance, or index sensitive "channel," where the learned weights may adapt to the precision of the various base classifiers. Another simple generalization results from using a nonuniform carrier density $q_0(\pi)$.

A further generalization is achieved by considering that for a given coding matrix, the trained classifier for a given column outputs either $\{y_i\}_{i \in I}|\{y_i\}_{i \in \bar{I}}$ or $\{y_i\}_{i \in \bar{I}}|\{y_i\}_{i \in I}$ depending on the input $x$. Allowing the output of the classifier instead to belong to other grades of $\mathfrak{W}_n$ results in a model that corresponds to error correcting output codes with non-binary codes. While this is somewhat antithetic to the original spirit of ECOC—reducing multiclass to binary—the base classifiers in ECOC are often multiclass classifiers such as decision trees in [4]. For such classifiers, the task instead can be viewed as reducing multiclass to partial ranking. Moreover, there need not be an explicit coding matrix. Instead, the input rankers may output different partial rankings for different inputs, which are then combined according to model (6). In this way, a different coding matrix is built for each example in a dynamic manner. Such a scheme may be attractive in bypassing the problem of designing the coding matrix.

# 6 Summary

An algebraic framework has been presented for classification and ranking, leading to conditional models on the ranking poset that are defined in terms of an invariant distance or dissimilarity function. Using the invariance properties of the distances, we derived a generative interpretation of the probabilistic model, which may prove to be useful in model selection and validation. Through different choices of the components $q_0, \Pi, \Sigma$ and $d$, the family of models was shown to include as special cases the Mallows $\phi$ model, and the classifier combination methods used by logistic models, boosting, cranking, and error correcting output codes. In the case of ECOC, the poset framework shows how probabilities may be assigned to partial rankings in a way that is consistent with the usual definitions of ECOC, and suggests several natural extensions.

## Acknowledgments

We thank D. Critchlow, G. Hulten and J. Verducci for helpful input on the paper. This work was supported in part by NSF grant CCR-0122581.

## References

[1] M. Collins, R. E. Schapire, and Y. Singer. Logistic regression, AdaBoost and Bregman distances. *Machine Learning*, 48, 2002.

[2] D. E. Critchlow. *Metric Methods for Analyzing Partially Ranked Data*. Lecture Notes in Statistics, volume 34, Springer, 1985.

[3] D. E. Critchlow and J. S. Verducci. Detecting a trend in paired rankings. *Journal of the Royal Statistical Society C*, 41(1):17–29, 1992.

[4] T. G. Dietterich and G. Bakiri. Solving multiclass learning problems via error-correcting codes. *Journal of Artificial Intelligence Research*, 2:263–286, 1995.

[5] P. D. Feigin. Modeling and analyzing paired ranking data. In M. A. Fligner and J. S. Verducci, editors, *Probability Models and Statistical Analyses for Ranking Data*. Springer, 1992.

[6] M. A. Fligner and J. S. Verducci. Posterior probabilities for a concensus ordering. *Psychometrika*, 55:53–63, 1990.

[7] Y. Freund and R. E. Schapire. Experiments with a new boosting algorithm. In *International Conference on Machine Learning*, 1996.

[8] M. G. Kendall. A new measure of rank correlation. *Biometrika*, 30, 1938.

[9] G. Lebanon and J. Lafferty. Boosting and maximum likelihood for exponential models. In *Advances in Neural Information Processing Systems, 15*, 2001.

[10] G. Lebanon and J. Lafferty. Cranking: Combining rankings using conditional probability models on permutations. In *International Conference on Machine Learning*, 2002.

[11] C. L. Mallows. Non-null ranking models. *Biometrika*, 44:114–130, 1957.

[12] R. P. Stanley. *Enumerative Combinatorics*, volume 1. Wadsworth & Brooks/Cole Mathematics Series, 1986.
